# Exploiting Model Uncertainty Estimates for Safe Dynamic Control Learning

**Jeff G. Schneider**
The Robotics Institute
Carnegie Mellon University
Pittsburgh, PA 15213
schneide@cs.cmu.edu

## Abstract

Model learning combined with dynamic programming has been shown to be effective for learning control of continuous state dynamic systems. The simplest method assumes the learned model is correct and applies dynamic programming to it, but many approximators provide uncertainty estimates on the fit. How can they be exploited? This paper addresses the case where the system must be prevented from having catastrophic failures during learning. We propose a new algorithm adapted from the dual control literature and use Bayesian locally weighted regression models with dynamic programming. A common reinforcement learning assumption is that aggressive exploration should be encouraged. This paper addresses the converse case in which the system has to reign in exploration. The algorithm is illustrated on a 4 dimensional simulated control problem.

## 1 Introduction

Reinforcement learning and related grid-based dynamic programming techniques are increasingly being applied to dynamic systems with continuous valued state spaces. Recent results on the convergence of dynamic programming methods when using various interpolation methods to represent the value (or cost-to-go) function have given a sound theoretical basis for applying reinforcement learning to continuous valued state spaces [Gordon, 1995]. These are important steps toward the eventual application of these methods to industrial learning and control problems.

It has also been reported recently that there are significant benefits in data and computational efficiency when data from running a system is used to build a model, rather than using it once for single value function updates (as Q-learning would do) and discarding it [Sutton, 1990, Moore and Atkeson, 1993, Schaal and Atkeson, 1993, Davies, 1996]. Dynamic programming sweeps can then be done on the learned model either off-line or on-line. In its vanilla form, this method assumes the model is correct and does deterministic dynamic programming using the model. This assumption is often not correct, especially in the early stages of learning. When learning simulated or software systems, there may be no harm in the fact that this

assumption does not hold. However, in real, physical systems there are often states that really are catastrophic and must be avoided even during learning. Worse yet, learning may have to occur during normal operation of the system in which case its performance during learning must not be significantly degraded.

The literature on adaptive and optimal linear control theory has explored this problem considerably under the names stochastic control and dual control. Overviews can be found in [Kendrick, 1981, Bar-Shalom and Tse, 1976]. The control decision is based on three components call the *deterministic, cautionary,* and *probing* terms. The deterministic term assumes the model is perfect and attempts to control for the best performance. Clearly, this may lead to disaster if the model is inaccurate. Adding a cautionary term yields a controller that considers the uncertainty in the model and chooses a control for the best expected performance. Finally, if the system learns while it is operating, there may be some benefit to choosing controls that are suboptimal and/or risky in order to obtain better data for the model and ultimately achieve better long-term performance. The addition of the probing term does this and gives a controller that yields the best long-term performance.

The advantage of dual control is that its strong mathematical foundation can provide the optimal learning controller under some assumptions about the system, the model, noise, and the performance criterion. Dynamic programming methods such as reinforcement learning have the advantage that they do not make strong assumptions about the system, or the form of the performance measure. It has been suggested [Atkeson, 1995, Atkeson, 1993] that techniques used in global linear control, including caution and probing, may also be applicable in the local case. In this paper we propose an algorithm that combines grid based dynamic programming with the *cautionary* concept from dual control via the use of a Bayesian locally weighted regression model.

Our algorithm is designed with industrial control applications in mind. A typical scenario is that a production line is being operated conservatively. There is data available from its operation, but it only covers a small region of the state space and thus can not be used to produce an accurate model over the whole potential range of operation. Management is interested in improving the line's response to changes in setpoints or disturbances, but can not risk much loss of production during the learning process. The goal of our algorithm is to collect new data and optimize the process while explicitly minimizing the risk.

## 2   The Algorithm

Consider a system whose dynamics are given by $x^{k+1} = f(x^k, u^k)$. The state, $x$, and control, $u$, are real valued vectors and $k$ represents discrete time increments. A model of $f$ is denoted as $\hat{f}$. The task is to minimize a cost functional of the form $J = \sum_{k=0}^{N} L(x^k, u^k, k)$ subject to the system dynamics. $N$ may or may not be fixed depending on the problem. $L$ is given, but $f$ must be learned. The goal is to acquire data to learn $f$ in order to minimize $J$ without incurring huge penalties in $J$ during learning. There is an implicit assumption that the cost function defines catastrophic states. If it were known that there were no disasters to avoid, then simpler, more aggressive algorithms would likely outperform the one presented here. The top level algorithm is as follows:

1. Acquire some data while operating the system from an existing controller.
2. Construct a model from the data using Bayesian locally weighted regression.
3. Perform DP with the model to compute a value function and a policy.
4. Operate the system using the new policy and record additional data.

5. Repeat to step 2 while there is still some improvement in performance.

In the rest of this section we describe steps 2 and 3.

## 2.1 Bayesian locally weighted regression

We use a form of locally weighted regression [Cleveland and Delvin, 1988, Atkeson, 1989, Moore, 1992] called Bayesian locally weighted regression [Moore and Schneider, 1995] to build a model from data. When a query, $x_q$, is made, each of the stored data points receives a weight $w_i = \exp(-\|x_i - x_q\|^2/K)$. $K$ is the *kernel width* which controls the amount of localness in the regression. For Bayesian LWR we assume a wide, weak normal-gamma prior on the coefficients of the regression model and the inverse of the noise covariance. The result of a prediction is a $t$ distribution on the output that remains well defined even in the absence of data (see [Moore and Schneider, 1995] and [DeGroot, 1970] for details).

The distribution of the prediction in regions where there is little data is crucial to the performance of the DP algorithm. As is often the case with learning through search and experimentation, it is at least as important that a function approximator predicts its own ignorance in regions of no data as it is how well it interpolates in data rich regions.

## 2.2 Grid based dynamic programming

In dynamic programming, the optimal value function, $V$, represents the cost-to-go from each state to the end of the task assuming that the optimal policy is followed from that point on. The value function can be computed iteratively by identifying the best action from each state and updating it according to the expected results of the action as given by a model of the system. The update equation is:

$$V^{k+1}(x) = \min_{u \in U} L(x, u) + V^k(\hat{f}(x, u)) \tag{1}$$

In our algorithm, updates to the value function are computed while considering the probability distribution on the results of each action. If we assume that the output of the real system at each time step is an independent random variable whose probability density function is given by the uncertainty from the model, the update equation is as follows:

$$V^{k+1}(x) = \min_{u \in U} L(x, u) + E[V^k(f(x, u))|\hat{f}] \tag{2}$$

Note that the independence assumption does not hold when reasonably smooth system dynamics are modeled by a smooth function approximator. The model error at one time step along a trajectory is highly correlated with the model error at the following step assuming a small distance traveled during the time step.

Our algorithm for DP with model uncertainty on a grid is as follows:

1. Discretize the state space, $X$, and the control space, $U$.

2. For each state and each control cache the cost of taking this action from this state. Also compute the probability density function on the next state from the model and cache the information. There are two cases which are shown graphically in fig. 1:

   - If the distribution is much narrower than the grid spacing, then the model is confident and a deterministic update will be done according to eq. 1. Multilinear interpolation is used to compute the value function at the mean of the predicted next state [Davies, 1996].

   - Otherwise, a stochastic update will be done according to eq. 2. The pdf of each of the state variables is stored, discretized at the same intervals as the grid representing the value function. Output independence is

High Confidence Next State          Low Confidence Next State

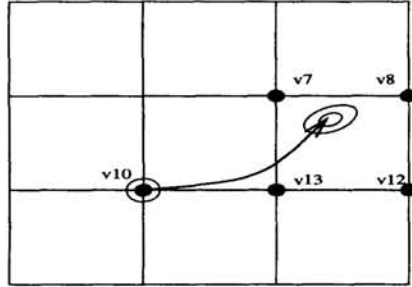
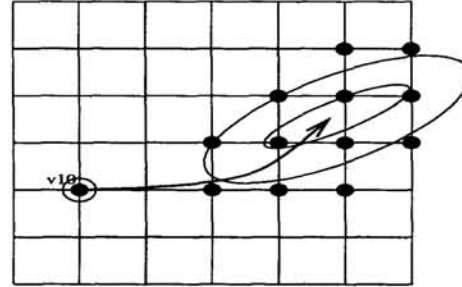

Figure 1: Illustration of the two kinds of cached updates. In the high confidence scenario the transition is treated as deterministic and the value function is computed with multilinear interpolation: $V_{10}^{k+1} = L(x, u) + 0.4V_7^k + 0.3V_8^k + 0.2V_{11}^k + 0.1V_{12}^k$. In the low confidence scenario the transition is treated stochastically and the update takes a weighted sum over all the vertices of significant weight as well as the probability mass outside the grid: $V_{10}^{k+1} = L(x, u) + \dfrac{\sum_{\{x'|p(x')>\epsilon\}} p(x'|\hat{f}, x, u)V^k(x')}{\sum_{\{x'|p(x')>\epsilon\}} p(x'|\hat{f}, x, u)}$.

assumed and later the pdf of each grid point will be computed as the product of the pdfs for each dimension and a weighted sum of all the grid points with significant weight will be computed. Also the total probability mass outside the bounds of the grid is computed and stored.

3. For each state, use the cached information to estimate the cost of choosing each action from that state. Update the value function at that state according to the cost of the best action found.

4. Repeat 3 until the value function converges, or the desired number of steps has been reached in finite step problems.

5. Record the best action (policy) for each grid point.

## 3   Experiments: Minimal Time Cart-Pole Maneuvers

The inverted pendulum is a well studied problem. It is easy to learn to stabilize it in a small number of trials, but not easy to learn quick maneuvers. We demonstrate our algorithm on the harder problem of moving the cart-pole stably from one position to another as quickly as possible. We assume we have a controller that can balance the pole and would like to learn to move the cart quickly to new positions, but never drop the pole during the learning process. The simulation equations and parameters are from [Barto et al., 1983] and the task is illustrated at the top of fig. 2. The state vector is $x = [$ pole angle $(\theta)$, pole angular velocity $(\dot{\theta})$, cart position $(\rho)$, cart velocity $(\dot{\rho})$ ]. The control vector, $u$, is the one dimensional force applied to the cart. $x_0$ is [0 0 17 0] and the cost function is $J = \sum_{i=0}^{N} x^T x + 0.01u^T u$. $N$ is not fixed. It is determined by the amount of time it takes for the system to reach a goal region about the target state, [0 0 0 0]. If the pole is dropped, the trial ends and an additional penalty of $10^6$ is incurred.

This problem has properties similar to familiar process control problems such as cooking, mixing, or cooling, because it is trivial to stabilize the system and it can be moved slowly to a new desired position while maintaining the stability by slowly changing positional setpoints. In each case, the goal is to learn how to respond faster without causing any disasters during, or after, the learning process.

### 3.1   Learning an LQR controller

We first learn a linear quadratic regulator that balances the pole. This can be done with minimal data. The system is operated from the state, [0 0 0 0] for 10 steps of length 0.1 seconds with a controller that chooses $u$ randomly from a zero mean gaussian with standard deviation 0.5. This is repeated to obtain a total of 20 data points. That data is used to fit a global linear model mapping $x$ onto $x'$. An LQR controller is constructed from the model and the given cost function following the derivation in [Dyer and McReynolds, 1970].

The resulting linear controller easily stabilizes the pole and can even bring the system stably (although very inefficiently as it passes through the goal several times before coming to rest there) to the origin when started as far out as $x = [0\ 0\ 10\ 0]$. If the cart is started further from the origin, the controller crashes the system.

### 3.2   Building the initial Bayesian LWR model

We use the LQR controller to generate data for an initial model. The system is started at $x = [0\ 0\ 1\ 0]$ and controlled by the LQR controller with gaussian noise added as before. The resulting 50 data points are stored for an LWR model that maps $[\theta, \dot{\theta}, u] \rightarrow [\ddot{\theta}, \ddot{p}]$. The data in each dimension of the state and control space is scaled to [0 1]. In this scaled space, the LWR kernel width is set to 1.0.

Next, we consider the deterministic DP method on this model. The grid covers the ranges: $[\pm1.0\ \pm4.0\ \pm21.0\ \pm20.0]$ and is discretized to [11 9 11 9] levels. The control is $\pm30.0$ discretized to 15 levels. Any state outside the grid bounds is considered failure and incurs the $10^6$ penalty. If we assume the model is correct, we can use deterministic DP on the grid to generate a policy. The computation is done with fixed size steps in time of 0.25 seconds. We observe that this policy is able to move the system safely from an initial state of [0 0 12 0], but crashes if it is started further out. Failure occurs because the best path generated using the model strays far from the region of the data (in variables $\theta$ and $\dot{\theta}$) used to construct the model.

It is disappointing that the use of LWR for nonlinear modeling didn't improve much over a globally linear model and an LQR controller. We believe this is a common situation. It is difficult to build better controllers from naive use of nonlinear modeling techniques because the available data models only a narrow region of operation and safely acquiring a wider range of data is difficult.

### 3.3   Cautionary dynamic programming

At this point we are ready to test our algorithm. Step 3 is executed using the LWR model from the data generated by the LQR controller as before. A trace of the system's operation when started at a distance of 17 from the goal is shown at the top of fig. 2. The controller is extremely conservative with respect to the angle of the pole. The pole is never allowed to go outside $\pm0.13$ radians. Even as the cart approaches the goal at a moderate velocity the controller chooses to overshoot the goal considerably rather than making an abrupt action to brake the system.

The data from this run is added to the model and the steps are repeated. Traces of the runs from three iterations of the algorithm are shown in fig. 2. At each trial, the controller becomes more aggressive and completes the task with less cost. After the third iteration, no significant improvement is observed. The costs are summarized and compared with the LQR and deterministic DP controllers in table 1.

Fig. 3 is another illustration of how the policy becomes increasingly aggressive. It plots the pole angle vs. the pole angular velocity for the original LQR data and the executions at each of the following three trials. In summary, our algorithm is able

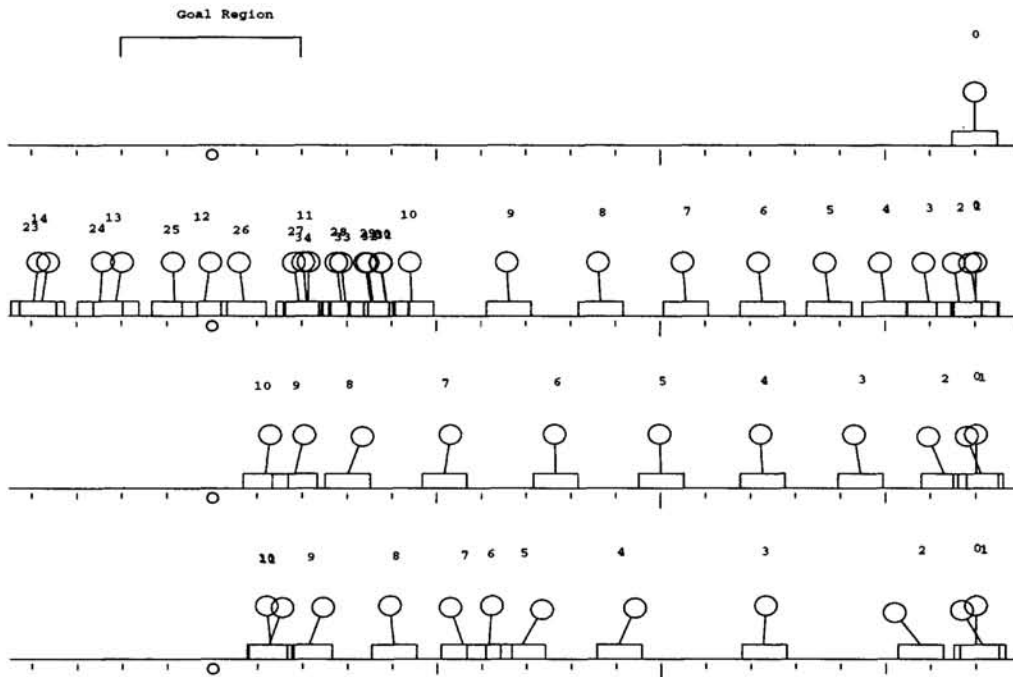

Figure 2: The task is to move the cart to the origin as quickly as possible without dropping the pole. The bottom three pictures show a trace of the policy execution obtained after one, two, and three trials (shown in increments of 0.5 seconds)

| Controller | Number of data points used to build the controller | Cost from initial state 17 |
|---|---|---|
| LQR | 20 | failure |
| Deterministic DP | 50 | failure |
| Stochastic DP trial 1 | 50 | 12393 |
| Stochastic DP trial 2 | 221 | 7114 |
| Stochastic DP trial 3 | 272 | 6270 |

Table 1: Summary of experimental results

to start from a simple controller that can stabilize the pole and learn to move it aggressively over a long distance without ever dropping the pole during learning.

## 4 Discussion

We have presented an algorithm that uses Bayesian locally weighted regression models with dynamic programming on a grid. The result is a cautionary adaptive control algorithm with the flexibility of a non-parametric nonlinear model instead of the more restrictive parametric models usually considered in the dual control literature. We note that this algorithm presents a viewpoint on the exploration vs exploitation issue that is different from many reinforcement learning algorithms, which are devised to encourage exploration (as in the *probing* concept in dual control). However, we argue that modeling the data first with a continuous function approximator and then doing DP on the model often leads to a situation where exploration must be *inhibited* to prevent disasters. This is particularly true in the case of real, physical systems.

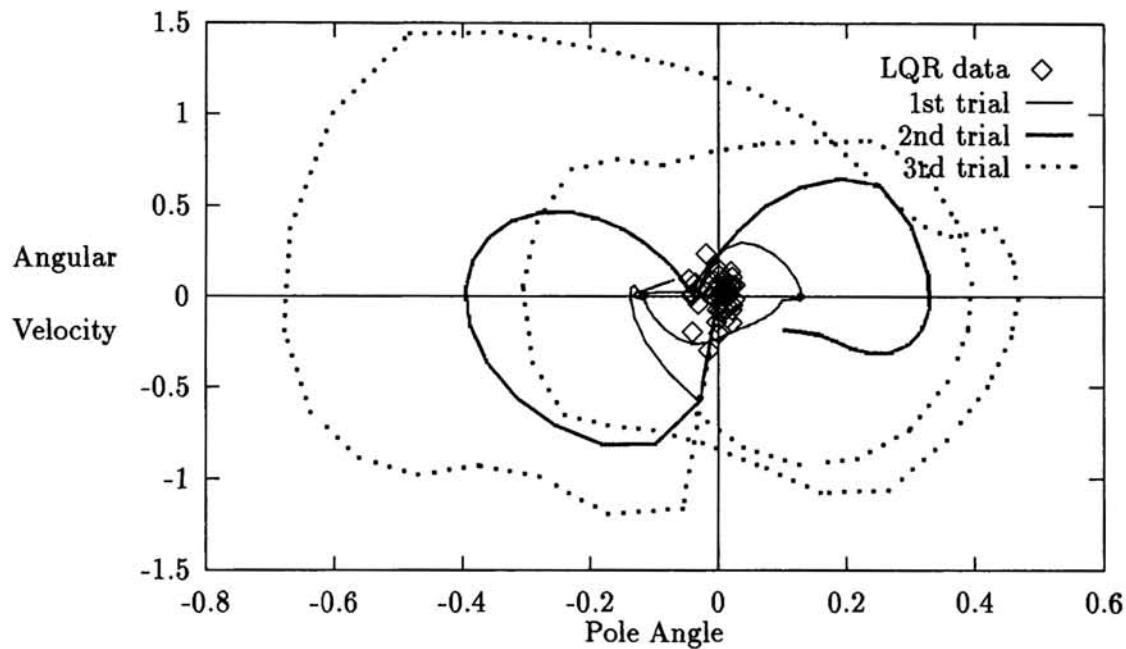

Figure 3: Execution trace. At each iteration, the controller is more aggressive.

# References

[Atkeson, 1989] C. Atkeson. Using local models to control movement. In *Advances in Neural Information Processing Systems*, 1989.

[Atkeson, 1993] C. Atkeson. Using local trajectory optimizers to speed up global optimization in dynamic programming. In *Advances in Neural Information Processing Systems (NIPS-6)*, 1993.

[Atkeson, 1995] C. Atkeson. Local methods for active learning. Invited talk at AAAI Fall Symposium on Active Learning, 1995.

[Bar-Shalom and Tse, 1976] Y. Bar-Shalom and E. Tse. *Concepts and Methods in Stochastic Control*. Academic Press, 1976.

[Barto et al., 1983] A. Barto, R. Sutton, and C. Anderson. Neuronlike adaptive elements that can solve difficult learning control problems. *IEEE Transactions on Systems, Man, and Cybernetics*, 1983.

[Cleveland and Delvin, 1988] W. Cleveland and S. Delvin. Locally weighted regression: An approach to regression analysis by local fitting. *Journal of the American Statistical Association*, pages 596–610, September 1988.

[Davies, 1996] S. Davies. Applying grid-based interpolation to reinforcement learning. In *Neural Information Processing Systems 9*, 1996.

[DeGroot, 1970] M. DeGroot. *Optimal Statistical Decisions*. McGraw-Hill, 1970.

[Dyer and McReynolds, 1970] P. Dyer and S. McReynolds. *The Computation and Theory of Optimal Control*. Academic Press, 1970.

[Gordon, 1995] G. Gordon. Stable function approximation in dynamic programming. In *The 12th International Conference on Machine Learning*, 1995.

[Kendrick, 1981] D. Kendrick. *Stochastic Control for Economic Models*. McGraw-Hill, 1981.

[Moore and Atkeson, 1993] A. Moore and C. Atkeson. Prioritized sweeping: Reinforcement learning with less data and less real time. *Machine Learning*, 13(1):103–130, 1993.

[Moore and Schneider, 1995] A. Moore and J. Schneider. Memory based stochastic optimization. In *Advances in Neural Information Processing Systems (NIPS-8)*, 1995.

[Moore, 1992] A. Moore. Fast, robust adaptive control by learning only forward models. In *Advances in Neural Information Processing Systems 4*, 1992.

[Schaal and Atkeson, 1993] S. Schaal and C. Atkeson. Assessing the quality of learned local models. In *Advances in Neural Information Processing Systems (NIPS-6)*, 1993.

[Sutton, 1990] R. Sutton. First results with dyna, an intergrated architecture for learning, planning, and reacting. In *AAAI Spring Symposium on Planning in Uncertain, Unpredictable, or Changing Environments*, 1990.
